# Near-Optimal MAP Inference
# for Determinantal Point Processes

**Jennifer Gillenwater     Alex Kulesza     Ben Taskar**
Computer and Information Science
University of Pennsylvania
{jengi,kulesza,taskar}@cis.upenn.edu

## Abstract

Determinantal point processes (DPPs) have recently been proposed as computationally efficient probabilistic models of diverse sets for a variety of applications, including document summarization, image search, and pose estimation. Many DPP inference operations, including normalization and sampling, are tractable; however, finding the most likely configuration (MAP), which is often required in practice for decoding, is NP-hard, so we must resort to approximate inference. This optimization problem, which also arises in experimental design and sensor placement, involves finding the largest principal minor of a positive semidefinite matrix. Because the objective is log-submodular, greedy algorithms have been used in the past with some empirical success; however, these methods only give approximation guarantees in the special case of monotone objectives, which correspond to a restricted class of DPPs. In this paper we propose a new algorithm for approximating the MAP problem based on continuous techniques for submodular function maximization. Our method involves a novel continuous relaxation of the log-probability function, which, in contrast to the multilinear extension used for general submodular functions, can be evaluated and differentiated exactly and efficiently. We obtain a practical algorithm with a 1/4-approximation guarantee for a more general class of non-monotone DPPs; our algorithm also extends to MAP inference under complex polytope constraints, making it possible to combine DPPs with Markov random fields, weighted matchings, and other models. We demonstrate that our approach outperforms standard and recent methods on both synthetic and real-world data.

## 1  Introduction

Informative subset selection problems arise in many applications where a small number of items must be chosen to represent or cover a much larger set; for instance, text summarization [1, 2], document and image search [3, 4, 5], sensor placement [6], viral marketing [7], and many others. Recently, probabilistic models extending determinantal point processes (DPPs) [8, 9] were proposed for several such problems [10, 5, 11]. DPPs offer computationally attractive properties, including exact and efficient computation of marginals [8], sampling [12, 5], and (partial) parameter estimation [13]. They are characterized by a notion of diversity, as shown in Figure 1; points in the plane sampled from a DPP (center) are more spread out than those sampled independently (left).

However, in many cases we would like to make use of the most likely configuration (MAP inference, right), which involves finding the largest principal minor of a positive semidefinite matrix. This is an NP-hard problem [14], and so we must resort to approximate inference methods. The DPP probability is a log-submodular function, and hence greedy algorithms are natural; however, the standard greedy algorithm of Nemhauser and Wolsey [15] offers an approximation guarantee of $1 - 1/e$ only for non-decreasing (monotone) submodular functions, and does not apply for general

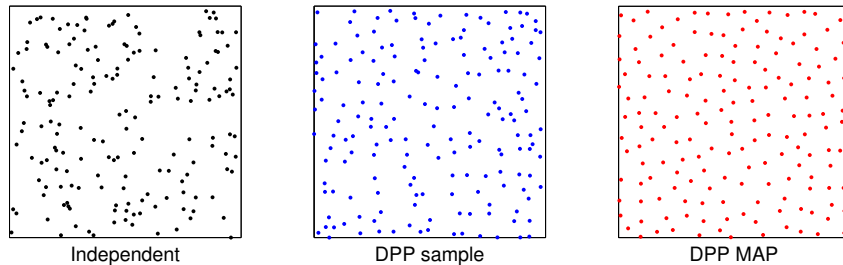

Figure 1: From left to right, a set of points in the plane sampled independently at random, a sample drawn from a DPP, and an approximation of the DPP MAP set estimated by our algorithm.

DPPs. In addition, we are are often interested in conditioning MAP inference on knapsack-type budget constraints, matroid constraints, or general polytope constraints. For example, we might consider a DPP model over edges of a bipartite graph and ask for the most likely set under the one-to-one matching constraint. In this paper we propose a new algorithm for approximating MAP inference that handles these types of constraints for non-monotone DPPs.

Recent work on non-monotone submodular function optimization can be broadly split into combinatorial versus continuous approaches. Among combinatorial methods, modified greedy, local search and simulated annealing algorithms provide certain constant factor guarantees [16, 17, 18] and have been recently extended to optimization under knapsack and matroid constraints [19, 20]. Continuous methods [21, 22] use a multilinear extension of the submodular set function to the convex hull of the feasible sets and then round fractional solutions obtained by maximizing in the interior of the polytope. Our algorithm falls into the continuous category, using a novel and efficient non-linear continuous extension specifically tailored to DPPs. In comparison to the constant-factor algorithms for general submodular functions, our approach is more efficient because we have explicit access to the objective function and its gradient. In contrast, general submodular functions assume a simple function oracle and need to employ sampling to estimate function and gradient values in the polytope interior. We show that our non-linear extension enjoys some of the critical properties of the standard multilinear extension and propose an efficient algorithm that can handle solvable polytope constraints. Our algorithm compares favorably to greedy and recent "symmetric" greedy [18] methods on unconstrained simulated problems, simulated problems under matching constraints, and a real-world matching task using quotes from political candidates.

## 2  Background

Determinantal point processes (DPPs) are distributions over subsets that prefer diversity. Originally, DPPs were introduced to model fermions in quantum physics [8], but since then they have arisen in a variety of other settings including non-intersecting random paths, random spanning trees, and eigenvalues of random matrices [9, 23, 12]. More recently, they have been applied as probabilistic models for machine learning problems [10, 13, 5, 11].

Formally, a DPP $\mathcal{P}$ on a set of items $\mathcal{Y} = \{1, 2, \ldots, N\}$ is a probability measure on $2^{\mathcal{Y}}$, the set of all subsets of $\mathcal{Y}$. For every $Y \subseteq \mathcal{Y}$ we have:

$$\mathcal{P}(Y) \propto \det(L_Y) \tag{1}$$

where $L$ is a positive semidefinite matrix. $L_Y \equiv [L_{ij}]_{i,j \in Y}$ denotes the restriction of $L$ to the entries indexed by elements of $Y$, and $\det(L_\emptyset) = 1$. If $L$ is written as a Gram matrix, $L = B^\top B$, then the quantity $\det(L_Y)$ can be interpreted as the squared volume spanned by the column vectors $B_i$ for $i \in Y$. If $L_{ij} = B_i^\top B_j$ is viewed as a measure of similarity between items $i$ and $j$, then when $i$ and $j$ are similar their vectors are relatively non-orthogonal, and therefore sets including both $i$ and $j$ will span less volume and be less probable. This is illustrated in Figure 2. As a result, DPPs assign higher probability to sets that are diverse under $L$.

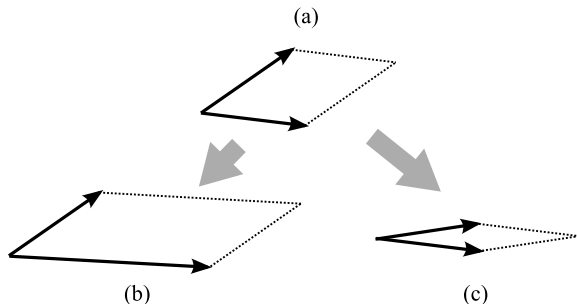

Figure 2: (a) The DPP probability of a set $Y$ depends on the volume spanned by vectors $B_i$ for $i \in Y$. (b) As length increases, so does volume. (c) As similarity increases, volume decreases.

The normalization constant in Equation (1) can be computed explicitly thanks to the identity

$$\sum_Y \det(L_Y) = \det(L + I) \,, \tag{2}$$

where $I$ is the $N \times N$ identity matrix. In fact, a variety of probabilistic inference operations can be performed efficiently, including sampling, marginalization, and conditioning [12, 24]. However, the *maximum a posteriori* (MAP) problem $\arg \max_Y \det(L_Y)$ is NP-hard [14]. In many practical situations it would be useful to approximate the MAP set; for instance, during decoding, online training, etc.

## 2.1   Submodularity

A function $f : 2^{\mathcal{Y}} \to \mathbb{R}$ is called submodular if it satisfies

$$f(X \cup \{i\}) - f(X) \geq f(Y \cup \{i\}) - f(Y) \tag{3}$$

whenever $X \subseteq Y$ and $i \notin Y$. Intuitively, the contribution made by a single item $i$ only decreases as the set grows. Common submodular functions include the mutual information of a set of variables and the number of cut edges leaving a set of vertices of a graph. A submodular function $f$ is called nondecreasing (or monotone) when $X \subseteq Y$ implies $f(X) \leq f(Y)$.

It is possible to show that $\log \det(L_Y)$ is a submodular function: entropy is submodular, and the entropy of a Gaussian is proportional to $\log \det(\Sigma_Y)$ (plus a linear term in $|Y|$), where $\Sigma$ is the covariance matrix. Submodular functions are easy to minimize, and a variety of algorithms exist for approximately maximizing them; however, to our knowledge none of these existing algorithms simultaneously allows for general polytope constraints on the set $Y$, offers an approximation guarantee, and can be implemented in practice without expensive sampling to approximate the objective. We provide a technique that addresses all three criteria for the DPP MAP problem, although approximation guarantees for the general polytope case depend on the choice of rounding algorithm and remain an open problem. We use the submodular maximization algorithm of [21] as a starting point.

## 3   MAP Inference

We seek an approximate solution to the generalized DPP MAP problem $\arg \max_{Y \in S} \log \det(L_Y)$, where $S \subseteq [0, 1]^N$ and $Y \in S$ means that the characteristic vector $\mathbb{I}(Y)$ is in $S$. We will assume that $S$ is a down-monotone, solvable polytope; down-monotone means that for $\boldsymbol{x}, \boldsymbol{y} \in [0, 1]^N$, $\boldsymbol{x} \in S$ implies $\boldsymbol{y} \in S$ whenever $\boldsymbol{x} \geq \boldsymbol{y}$ (that is, whenever $x_i \geq y_i \; \forall i$), and solvable means that for any linear objective function $g(\boldsymbol{x}) = \boldsymbol{a}^\top \boldsymbol{x}$, we can efficiently find $\boldsymbol{x} \in S$ maximizing $g(\boldsymbol{x})$.

One common approach for approximating discrete optimization problems is to replace the discrete variables with continuous analogs and extend the objective function to the continuous domain. When the resulting continuous optimization is solved, the result may include fractional variables. Typically, a rounding scheme is then used to produce a valid integral solution. As we will detail below,

we use a novel non-linear continuous relaxation that has a nice property: when the polytope is unconstrained, $S = [0,1]^N$, our method will (essentially) always produce integral solutions. For more complex polytopes, a rounding procedure is required.

When the objective $f(Y)$ is a submodular set function, as in our setting, the multilinear extension can be used to obtain certain theoretical guarantees for the relaxed optimization scheme described above [21, 25]. The multilinear extension is defined on a vector $\boldsymbol{x} \in [0,1]^N$:

$$F(\boldsymbol{x}) = \sum_Y \prod_{i \in Y} x_i \prod_{i \notin Y} (1 - x_i) f(Y) . \tag{4}$$

That is, $F(\boldsymbol{x})$ is the expected value of $f(Y)$ when $Y$ is the random set obtained by including element $i$ with probability $x_i$. Unfortunately, this expectation generally cannot be computed efficiently, since it involves summing over exponentially many sets $Y$. Thus, to use the multilinear extension in practice requires estimating its value and derivative via Monte Carlo techniques. This makes the optimization quite computationally expensive, as well as introducing a variety of technical convergence issues.

Instead, for the special case of DPP probabilities we propose a new continuous extension that is efficiently computable and differentiable. We refer to the following function as the softmax extension:

$$\tilde{F}(\boldsymbol{x}) = \log \sum_Y \prod_{i \in Y} x_i \prod_{i \notin Y} (1 - x_i) \exp(f(Y)) . \tag{5}$$

See the supplementary material for a visual comparison of Equations (4) and (5). While the softmax extension also involves a sum over exponentially many sets $Y$, we have the following theorem.

**Theorem 1.** *For a positive semidefinite matrix $L$ and $\boldsymbol{x} \in [0,1]^N$,*

$$\sum_Y \prod_{i \in Y} x_i \prod_{i \notin Y} (1 - x_i) \det(L_Y) = \det(\mathrm{diag}(\boldsymbol{x})(L - I) + I) . \tag{6}$$

All proofs are included in the supplementary material.

**Corollary 2.** *For $f(Y) = \log \det(L_Y)$, we have $\tilde{F}(\boldsymbol{x}) = \log \det(\mathrm{diag}(\boldsymbol{x})(L - I) + I)$ and*

$$\frac{\partial}{\partial x_i} \tilde{F}(\boldsymbol{x}) = \mathrm{tr}((\mathrm{diag}(\boldsymbol{x})(L - I) + I)^{-1}(L - I)_i) , \tag{7}$$

*where $(L - I)_i$ denotes the matrix obtained by zeroing all except the $i$th row of $L - I$.*

Corollary 2 says that softmax extension for the DPP MAP problem is computable and differentiable in $O(N^3)$ time. Using a variant of gradient ascent (Section 3.1), this will be sufficient to efficiently find a local maximum of the softmax extension over an arbitrary solvable polytope. It then remains to show that this local maximum comes with approximation guarantees.

## 3.1 Conditional gradient

When the optimization polytope $S$ is simple—for instance, the unit cube $[0,1]^N$—we can apply generic gradient-based optimization methods like L-BFGS to rapidly find a local maximum of the softmax extension. In situations where we are able to efficiently project onto the polytope $S$, we can apply projected gradient methods. In the general case, however, we assume only that the polytope is solvable. In such settings, we can use the conditional gradient algorithm (also known as the Frank-Wolfe algorithm) [26, 27]. Algorithm 1 describes the procedure; intuitively, at each step we move to a convex combination of the current point and the point maximizing the linear approximation of the function given by the current gradient. This ensures that we move in an increasing direction while remaining in $S$. Note that finding $\boldsymbol{y}$ requires optimizing a linear function over $S$; this step is efficient whenever the polytope is solvable.

## 3.2 Approximation bound

In order to obtain an approximation bound for the DPP MAP problem, we consider the two-phase optimization in Algorithm 2, originally proposed in [21]. The second call to LOCAL-OPT is necessary in theory; however, in practice it can usually be omitted with minimal loss (if any). We will show that Algorithm 2 produces a 1/4-approximation.

**Algorithm 1** LOCAL-OPT

> **Input:** function $\tilde{F}$, polytope $S$
> $\boldsymbol{x} \leftarrow \boldsymbol{0}$
> **while** not converged **do**
> > $\boldsymbol{y} \leftarrow \arg\max_{\boldsymbol{y}' \in S} \nabla \tilde{F}(\boldsymbol{x})^\top \boldsymbol{y}'$
> > $\alpha \leftarrow \arg\max_{\alpha' \in [0,1]} \tilde{F}(\alpha' \boldsymbol{x} + (1-\alpha')\boldsymbol{y})$
> > $\boldsymbol{x} \leftarrow \alpha \boldsymbol{x} + (1-\alpha)\boldsymbol{y}$
> **end while**
> **Output:** $\boldsymbol{x}$

**Algorithm 2** Approximating the DPP MAP

> **Input:** kernel $L$, polytope $S$
> Let $\tilde{F}(\boldsymbol{x}) = \log\det(\operatorname{diag}(\boldsymbol{x})(L-I)+I)$
> $\boldsymbol{x} \leftarrow$ LOCAL-OPT$(\tilde{F}, S)$
> $\boldsymbol{y} \leftarrow$ LOCAL-OPT$(\tilde{F}, S \cap \{\boldsymbol{y}' \mid \boldsymbol{y}' \leq \boldsymbol{1} - \boldsymbol{x}\})$
>
> **Output:** $\begin{cases} \boldsymbol{x} & : \tilde{F}(\boldsymbol{x}) > \tilde{F}(\boldsymbol{y}) \\ \boldsymbol{y} & : \text{otherwise} \end{cases}$

We begin by proving that the continuous extension $\tilde{F}$ is concave in positive directions, although it is not concave in general.

**Lemma 3.** *When $\boldsymbol{u}, \boldsymbol{v} \geq \boldsymbol{0}$, we have*

$$\frac{\partial^2}{\partial s \partial t} \tilde{F}(\boldsymbol{x} + s\boldsymbol{u} + t\boldsymbol{v}) \leq 0 \tag{8}$$

*wherever $\boldsymbol{0} < \boldsymbol{x} + s\boldsymbol{u} + t\boldsymbol{v} < \boldsymbol{1}$.*

**Corollary 4.** *$\tilde{F}(\boldsymbol{x} + t\boldsymbol{v})$ is concave along any direction $\boldsymbol{v} \geq \boldsymbol{0}$ (equivalently, $\boldsymbol{v} \leq \boldsymbol{0}$).*

Corollary 4 tells us that a local optimum $\boldsymbol{x}$ of $\tilde{F}$ has certain global properties—namely, that $\tilde{F}(\boldsymbol{x}) \geq \tilde{F}(\boldsymbol{y})$ whenever $\boldsymbol{y} \leq \boldsymbol{x}$ or $\boldsymbol{y} \geq \boldsymbol{x}$. This leads to the following result from [21].

**Lemma 5.** *If $\boldsymbol{x}$ is a local optimum of $\tilde{F}(\cdot)$, then for any $\boldsymbol{y} \in [0,1]^N$,*

$$2\tilde{F}(\boldsymbol{x}) \geq \tilde{F}(\boldsymbol{x} \vee \boldsymbol{y}) + \tilde{F}(\boldsymbol{x} \wedge \boldsymbol{y}) , \tag{9}$$

*where $(\boldsymbol{x} \vee \boldsymbol{y})_i = \max(x_i, y_i)$ and $(\boldsymbol{x} \wedge \boldsymbol{y})_i = \min(x_i, y_i)$.*

Following [21], we now define a surrogate function $\tilde{F}^*$. Let $\mathcal{X}_i \subseteq [0,1]$ be a subset of the unit interval representing $x_i = |\mathcal{X}_i|$, where $|\mathcal{X}_i|$ denotes the measure of $\mathcal{X}_i$. (Note that this representation is overcomplete, since there are in general many subsets of $[0,1]$ with measure $x_i$.) $\tilde{F}^*$ is defined on $\mathcal{X} = (\mathcal{X}_1, \mathcal{X}_2, \ldots, \mathcal{X}_N)$ by

$$\tilde{F}^*(\mathcal{X}) = \tilde{F}(\boldsymbol{x}), \quad \boldsymbol{x} = (|\mathcal{X}_1|, |\mathcal{X}_2|, \ldots, |\mathcal{X}_N|) . \tag{10}$$

**Lemma 6.** *$\tilde{F}^*$ is submodular.*

Lemmas 5 and 6 suffice to prove the following theorem, which appears for the multilinear extension in [21], bounding the approximation ratio of Algorithm 2.

**Theorem 7.** *Let $\tilde{F}(\boldsymbol{x})$ be the softmax extension of a nonnegative submodular function $f(Y) = \log\det(L_Y)$, let $\mathrm{OPT} = \max_{\boldsymbol{x}' \in S} \tilde{F}(\boldsymbol{x}')$, and let $\boldsymbol{x}$ and $\boldsymbol{y}$ be local optima of $\tilde{F}$ in $S$ and $S \cap \{\boldsymbol{y}' \mid \boldsymbol{y}' \leq \boldsymbol{1} - \boldsymbol{x}\}$, respectively. Then*

$$\max(\tilde{F}(\boldsymbol{x}), \tilde{F}(\boldsymbol{y})) \geq \frac{1}{4}\mathrm{OPT} \geq \frac{1}{4} \max_{Y \in S} \log\det(L_Y) . \tag{11}$$

Note that the softmax extension is an upper bound on the multilinear extension, thus Equation (11) is at least as tight as the corresponding result in [21].

**Corollary 8.** *Algorithm 2 yields a 1/4-approximation to the DPP MAP problem whenever $\log\det(L_Y) \geq 0$ for all $Y$. In general, the objective value obtained by Algorithm 2 is bounded below by $\frac{1}{4}(\mathrm{OPT} - p_0) + p_0$, where $p_0 = \min_Y \log\det(L_Y)$.*

In practice, filtering of near-duplicates can be used to keep $p_0$ from getting too small; however, in our empirical tests $p_0$ did not seem to have a significant effect on approximation quality.

### 3.3 Rounding

When the polytope $S$ is unconstrained, it is easy to show that the results of Algorithm 1—and, in turn, Algorithm 2—are integral (or can be rounded without loss).

**Theorem 9.** *If $S = [0, 1]^N$, then for any local optimum $\boldsymbol{x}$ of $\tilde{F}$, either $\boldsymbol{x}$ is integral or at least one fractional coordinate $x_i$ can be set to 0 or 1 without lowering the objective.*

More generally, however, the polytope $S$ can be complex, and the output of Algorithm 2 needs to be rounded. We speculate that the contention resolution rounding schemes proposed in [21] for the multilinear extension $F$ may be extensible to $\tilde{F}$, but do not attempt to prove so here. Instead, in our experiments we apply pipage rounding [28] and threshold rounding (rounding all coordinates up or down using a single threshold), which are simple and seem to work well in practice.

### 3.4 Model combination

In addition to theoretical guarantees and the empirical advantages we demonstrate in Section 4, the proposed approach to the DPP MAP problem offers a great deal of flexibility. Since the general framework of continuous optimization is widely used in machine learning, this technique allows DPPs to be easily combined with other models. For instance, if $S$ is the local polytope for a Markov random field, then, augmenting the objective with the (linear) log-likelihood of the MRF—additive linear objective terms do not affect the lemmas proved above—we can approximately compute the MAP configuration of the DPP-MRF product model. We might in this way model diverse objects placed in a sequence, or fit to an underlying signal like an image. Empirical studies of these possibilities are left to future work.

## 4 Experiments

To illustrate the proposed method, we compare it to the widely used greedy algorithm of Nemhauser and Wolsey [15] (Algorithm 3) and the recently proposed deterministic "symmetric" greedy algorithm [18], which has a $1/3$ approximation guarantee for unconstrained non-monotone problems. Note that, while a naive implementation of the $\arg\max$ in Algorithm 3 requires evaluating the objective for each item in $U$, here we can exploit the fact that DPPs are closed under conditioning to compute all necessary values with only two matrix inversions [5]. We report baseline runtimes using this optimized greedy algorithm, which is about 10 times faster than the naive version at $N = 200$. The code and data for all experiments can be downloaded from `http://www.seas.upenn.edu/~jengi/dpp-map.html`.

### 4.1 Synthetic data

As a first test, we approximate the MAP configuration for DPPs with random kernels drawn from a Wishart distribution. Specifically, we choose $L = B^\top B$, where $B \in \mathbb{R}^{N \times N}$ has entries drawn independently from the standard normal distribution, $b_{ij} \sim \mathcal{N}(0, 1)$. This results in $L \sim \mathcal{W}_N(N, I)$, a Wishart distribution with $N$ degrees of freedom and an identity covariance matrix. This distribution has several desirable properties: (1) in terms of eigenvectors, it spreads its mass uniformly over all unitary matrices [29], and (2) the probability density of eigenvalues $\lambda_1, \ldots, \lambda_N$ is

$$\exp\left(-\sum_{i=1}^{N} \lambda_i\right) \prod_{i=1}^{N} \frac{\prod_{j=i+1}^{N} (\lambda_i - \lambda_j)^2}{((N-i)!)^2} \,, \tag{12}$$

the first term of which deters the eigenvalues from being too large, and the second term of which encourages the eigenvalues to be well-separated [30]. Property (1) implies that we will see a variety of eigenvectors, which play an important role in the structure of a DPP [5]. Property (2) implies that interactions between these eigenvectors will be important, as no one eigenvalue is likely to dominate. Combined, these properties suggest that samples should encompass a wide range of DPPs.

Figure 3a shows performance results on these random kernels in the unconstrained setting. Our proposed algorithm outperforms greedy in general, and the performance gap tends to grow with the size of the ground set, $N$. (We let $N$ vary in the range $[50, 200]$ since prior work with DPPs

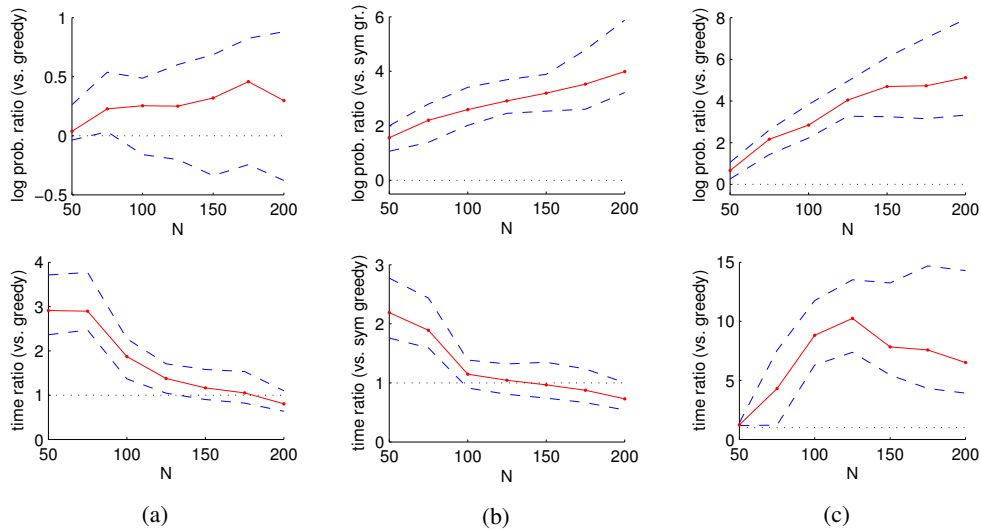

Figure 3: Median and quartile log probability ratios (top) and running time ratios (bottom) for 100 random trials. (a) The proposed algorithm versus greedy on unconstrained problems. (b) The proposed algorithm versus symmetric greedy on unconstrained problems. (c) The proposed algorithm versus greedy on constrained problems. Dotted black lines indicate equal performance.

in real-world scenarios [5, 13] has typically operated in this range.) Moreover, Figure 3a (bottom) illustrates that our method is of comparable efficiency at medium $N$, and becomes more efficient as $N$ grows. Despite the fact that the symmetric greedy algorithm [18] has an improved approximation guarantee of $1/3$, essentially the same analysis applies to Figure 3b.

Figure 3c summarizes the performance of our algorithm in a constrained setting. To create plausible constraints, in this setting we generate two separate random matrices $B^{(1)}$ and $B^{(2)}$, and then select random pairs of rows $(B_i^{(1)}, B_j^{(2)})$. Averaging $(B_i^{(1)} + B_j^{(2)})/2$ creates one row of the matrix $B$; we then set $L = B^\top B$. The constraints require that if $x_k$ corresponding to the $(i, j)$ pair is 1, no other $x_{k'}$ can have first element $i$ or second element $j$; i.e., the pairs cannot overlap. Since exact duplicate pairs produce identical rows in $L$, they are never both selected and can be pruned ahead of time. This means our constraints are of a form that allows us to apply pipage rounding to the possibly fractional result. Figure 3c shows even greater gains over greedy in this setting; however, enforcing the constraints precludes using fast methods like L-BFGS, so our optimization procedure is in this case somewhat slower than greedy.

## 4.2 Matched summarization

Finally, we demonstrate our approach using real-world data. Consider the following task: given a set of documents, select a set of document *pairs* such that the two elements within a pair are similar, but the overall set of pairs is diverse. For instance, we might want to compare the opinions of various authors on a range of topics—or even to compare the statements made at different points in time by the same author, e.g., a politician believed to have changed positions on various issues.

In this vein, we extract all the statements made by the eight main contenders in the 2012 US Republican primary debates: Bachmann, Cain, Gingrich, Huntsman, Paul, Perry, Romney, and Santorum. See the supplementary material for an example of some of these statements. Each pair of candidates $(a, b)$ constitutes one instance of our task. The task output is a set of statement pairs where the first statement in each pair comes from candidate $a$ and the second from candidate $b$. The goal of optimization is to find a set that is diverse (contains many topics, such as healthcare, foreign policy, immigration, etc.) but where both statements in each pair are topically similar.

Before formulating a DPP objective for this task, we perform some pre-processing. We filter short statements, leaving us with an average of 179 quotes per candidate (min = 93, max = 332 quotes).

**Algorithm 3** Greedy MAP for DPPs
___

**Input:** kernel $L$, polytope $S$
$Y \leftarrow \emptyset, U \leftarrow \mathcal{Y}$
**while** $U$ is not empty **do**
    $i^* \leftarrow \arg\max_{i \in U} \log\det(L_{Y \cup \{i\}})$
    **if** $\log\det(L_{Y \cup \{i^*\}}) < \log\det(L_Y)$
    **then**
        **break**
    **end if**
    $Y \leftarrow Y \cup \{i^*\}$
    $U \leftarrow \{i \mid i \notin Y, \mathbb{I}(Y \cup \{i\}) \in S\}$
**end while**
**Output:** $Y$
___

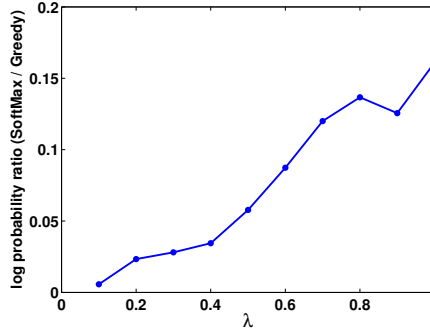

Figure 4: Log ratio of the objective value achieved by our method to that achieved by greedy for ten settings of match weight $\lambda$.

We parse the quotes, keeping only nouns. We further filter nouns by document frequency, keeping only those that occur in at least 10% of the quotes. Then we generate a feature matrix $W$ where $W_{qt}$ is the number of times term $t$ appears in quote $q$. This matrix is then normalized so that $\|W_q\|_2 = 1$, where $W_q$ is the $q$th row of $W$. For a given pair of candidates $(a, b)$ we compute the quality of each possible quote pair $(q_i^{(a)}, q_j^{(b)})$ as the dot product of their rows in $W$. While the model will naturally ignore low-quality pairs, for efficiency we throw away such pairs in pre-processing. For each of candidate $a$'s quotes $q_i^{(a)}$ we keep a pair with quote $j = \arg\max_{j'} \text{quality}(q_i^{(a)}, q_{j'}^{(b)})$ from candidate $b$, and vice-versa. The scores of the unpruned quotes, which we denote $\mathbf{r}$, are re-normalized to span the $[0, 1]$ range. To create a feature vector describing each pair, we simply add the corresponding pair of quote feature vectors and re-normalize, forming a new $W$ matrix.

Our task is to select some high-quality representative subset of the unpruned quote pairs. We formulate this as a DPP objective with kernel $L = MSM$, where $S_{ij}$ is a measurement of similarity between quote pairs $i$ and $j$, and $M$ is a diagonal matrix with $M_{ii}$ representing the match quality of pair $i$. We set $S = WW^T$ and $\text{diag}(M) = \sqrt{\exp(\lambda \mathbf{r})}$, where $\lambda$ is a hyperparameter. Large $\lambda$ places more emphasis on picking high-quality pairs than on making the overall set diverse.

To help limit the number of pairs selected when optimizing the objective, we add some constraints. For each candidate we cluster their quotes using $k$-means on the word feature vectors and impose the constraint that no more than one quote per cluster can be selected. We round the final solution using the threshold rounding scheme described in Section 3.3.

Figure 4 shows the result of optimizing this constrained objective, averaged over all 56 candidate pairs. For all settings of $\lambda$ we outperform greedy. In general, we observe that our algorithm is most improved compared to greedy when the constraints are in play. In this case, when $\lambda$ is small the constraints are less relevant, since the model has an intrinsic preference for smaller sets. On the other hand, when $\lambda$ is very large the algorithms must choose as many pairs as possible in order to maximize their score; in this case the constraints play an important role.

## 5 Conclusion

We presented a new approach to solving the MAP problem for DPPs based on continuous algorithms for submodular maximization. Unlike the multilinear extension used in the general case, the softmax extension we propose is efficiently computable and differentiable. Furthermore, it allows for general solvable polytope constraints, and yields a guaranteed 1/4-approximation in a subclass of DPPs. Our method makes it easy to combine DPPs with other models like MRFs or matching models, and is faster and more reliable than standard greedy methods on synthetic and real-world problems.

### Acknowledgments

This material is based upon work supported under a National Science Foundation Graduate Research Fellowship, Sloan Research Fellowship, and NSF Grant 0803256.

# References

[1] A. Nenkova, L. Vanderwende, and K. McKeown. A Compositional Context-Sensitive Multi-Document Summarizer: Exploring the Factors that Influence Summarization. In *Proc. SIGIR*, 2006.

[2] H. Lin and J. Bilmes. Multi-document Summarization via Budgeted Maximization of Submodular Functions. In *Proc. NAACL/HLT*, 2010.

[3] F. Radlinski, R. Kleinberg, and T. Joachims. Learning Diverse Rankings with Multi-Armed Bandits. In *Proc. ICML*, 2008.

[4] Y. Yue and T. Joachims. Predicting Diverse Subsets Using Structural SVMs. In *Proc. ICML*, 2008.

[5] A. Kulesza and B. Taskar. k-DPPs: Fixed-Size Determinantal Point Processes. In *Proc. ICML*, 2011.

[6] C. Guestrin, A. Krause, and A. Singh. Near-Optimal Sensor Placements in Gaussian Processes. In *Proc. ICML*, 2005.

[7] D. Kempe, J. Kleinberg, and E. Tardos. Influential Nodes in a Diffusion Model for Social Networks. In *Automata, Languages and Programming*, volume 3580 of *Lecture Notes in Computer Science*. 2005.

[8] O. Macchi. The Coincidence Approach to Stochastic Point Processes. *Advances in Applied Probability*, 7(1), 1975.

[9] D. Daley and D. Vere-Jones. *An Introduction to the Theory of Point Processes: Elementary Theory and Methods*. 2003.

[10] A. Kulesza and B Taskar. Structured Determinantal Point Processes. In *Proc. NIPS*, 2010.

[11] A. Kulesza, J. Gillenwater, and B. Taskar. Discovering Diverse and Salient Threads in Document Collections. In *Proc. EMNLP*, 2012.

[12] J. Hough, M. Krishnapur, Y. Peres, and B. Virág. Determinantal Processes and Independence. *Probability Surveys*, 3, 2006.

[13] A. Kulesza and B. Taskar. Learning Determinantal Point Processes. In *Proc. UAI*, 2011.

[14] C. Ko, J. Lee, and M. Queyranne. An Exact Algorithm for Maximum Entropy Sampling. *Operations Research*, 43(4), 1995.

[15] G. Nemhauser, L. Wolsey, and M. Fisher. An Analysis of Approximations for Maximizing Submodular Set Functions I. *Mathematical Programming*, 14(1), 1978.

[16] U. Feige, V. Mirrokni, and J. Vondrak. Maximizing Non-Monotone Submodular Functions. In *Proc. FOCS*, 2007.

[17] T. Robertazzi and S. Schwartz. An Accelerated Sequential Algorithm for Producing D-optimal Designs. *SIAM J. Sci. Stat. Comput.*, 10(2), 1989.

[18] N. Buchbinder, M. Feldman, J. Naor, and R. Schwartz. A Tight Linear Time (1/2)-Approximation for Unconstrained Submodular Maximization. In *Proc. FOCS*, 2012.

[19] A. Gupta, A. Roth, G. Schoenebeck, and K. Talwar. Constrained Nonmonotone Submodular Maximization: Offline and Secretary Algorithms. In *Internet and Network Economics*, volume 6484 of *LNCS*. 2010.

[20] S. Gharan and J. Vondrák. Submodular Maximization by Simulated Annealing. In *Proc. Soda*, 2011.

[21] C. Chekuri, J. Vondrák, and R. Zenklusen. Submodular Function Maximization via the Multilinear Relaxation and Contention Resolution Schemes. *arXiv:1105.4593*, 2011.

[22] M. Feldman, J. Naor, and R. Schwartz. Nonmonotone Submodular Maximization via a Structural Continuous Greedy Algorithm. *Automata, Languages and Programming*, 2011.

[23] A. Borodin and A. Soshnikov. Janossy Densities I. Determinantal Ensembles. *Journal of Statistical Physics*, 113(3), 2003.

[24] A. Borodin. Determinantal Point Processes. *arXiv:0911.1153*, 2009.

[25] M. Feldman, J. Naor, and R. Schwartz. A Unified Continuous Greedy Algorithm for Submodular Maximization. In *Proc. FOCS*, 2011.

[26] D. Bertsekas. *Nonlinear Programming*. Athena Scientific, 1999.

[27] M. Frank and P. Wolfe. An Algorithm for Quadratic Programming. *Naval Research Logistics Quarterly*, 3(1-2), 1956.

[28] A. Ageev and M. Sviridenko. Pipage Rounding: A New Method of Constructing Algorithms with Proven Performance Guarantee. *Journal of Combinatorial Optimization*, 8(3), 2004.

[29] A. James. Distributions of Matrix Variates and Latent Roots Derived from Normal Samples. *Annals of Mathematical Statistics*, 35(2), 1964.

[30] P. Hsu. On the Distribution of Roots of Certain Determinantal Equations. *Annals of Eugenics*, 9(3), 1939.

